# Learning to Make Coherent Predictions in Domains with Discontinuities

**Suzanna Becker and Geoffrey E. Hinton**
Department of Computer Science, University of Toronto
Toronto, Ontario, Canada M5S 1A4

## Abstract

We have previously described an unsupervised learning procedure that discovers spatially coherent properties of the world by maximizing the information that parameters extracted from different parts of the sensory input convey about some common underlying cause. When given random dot stereograms of curved surfaces, this procedure learns to extract surface depth because that is the property that is coherent across space. It also learns how to interpolate the depth at one location from the depths at nearby locations (Becker and Hinton, 1992). In this paper, we propose two new models which handle surfaces with discontinuities. The first model attempts to detect cases of discontinuities and reject them. The second model develops a mixture of expert interpolators. It learns to detect the locations of discontinuities and to invoke specialized, asymmetric interpolators that do not cross the discontinuities.

## 1 Introduction

Standard backpropagation is implausible as a model of perceptual learning because it requires an external teacher to specify the desired output of the network. We have shown (Becker and Hinton, 1992) how the external teacher can be replaced by internally derived teaching signals. These signals are generated by using the assumption that different parts of the perceptual input have common causes in the external world. Small modules that look at separate but related parts of the perceptual input discover these common causes by striving to produce outputs that agree with each other (see Figure 1 a). The modules may look at different modalities (e.g. vision and touch), or the same modality at different times (e.g. the consecutive 2-D views of a rotating 3-D object), or even spatially adjacent parts of the same image. In previous work, we showed that when our learning procedure is applied

to adjacent patches of 2-dimensional images, it allows a neural network that has no prior knowledge of the third dimension to discover depth in random dot stereograms of curved surfaces. A more general version of the method allows the network to discover the best way of interpolating the depth at one location from the depths at nearby locations. We first summarize this earlier work, and then introduce two new models which allow coherent predictions to be made in the presence of discontinuities.

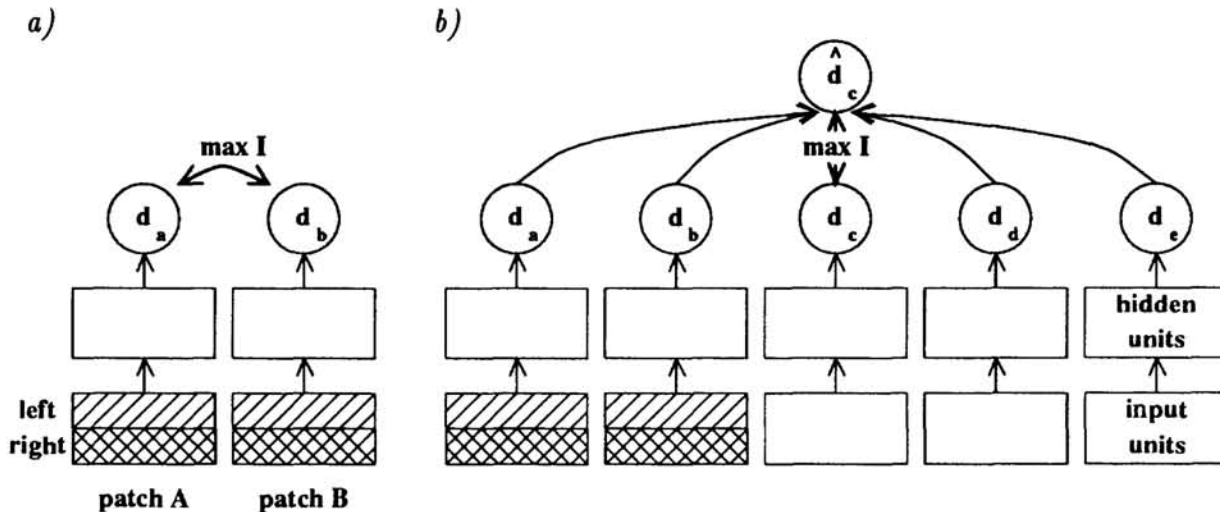

Figure 1: *a) Two modules that receive input from corresponding parts of stereo images. The first module receives input from stereo patch A, consisting of a horizontal strip from the left image (striped) and a corresponding strip from the right image (hatched). The second module receives input from an adjacent stereo patch B. The modules try to make their outputs, $d_a$ and $d_b$, convey as much information as possible about some underlying signal (i.e., the depth) which is common to both patches. b) The architecture of the interpolating network, consisting of multiple copies of modules like those in a) plus a layer of interpolating units. The network tries to maximize the information that the locally extracted parameter $d_c$ and the contextually predicted parameter $\hat{d}_c$ convey about some common underlying signal. We actually used 10 modules and the central 6 modules tried to maximize agreement between their outputs and contextually predicted values. We used weight averaging to constrain the interpolating function to be identical for all modules.*

## 2  Learning spatially coherent features in images

The simplest way to get the outputs of two modules to agree is to use the squared difference between the outputs as a cost function, and to adjust the weights in each module so as to minimize this cost. Unfortunately, this usually causes each module to produce the same constant output that is unaffected by the input to the module and therefore conveys no information about it. What we want is for the outputs of two modules to agree closely (i.e. to have a small expected squared difference) *relative* to how much they both vary as the input is varied. When this happens, the two modules must be responding to something that is common to their two inputs. In the special case when the outputs, $d_a$, $d_b$, of the two modules are scalars, a good

measure of agreement is:

$$I = 0.5 \log \frac{V(d_a + d_b)}{V(d_a - d_b)} \tag{1}$$

where $V$ is the variance over the training cases. If $d_a$ and $d_b$ are both versions of the same underlying Gaussian signal that have been corrupted by independent Gaussian noise, it can be shown that $I$ is the mutual information between the underlying signal and the average of $d_a$ and $d_b$. By maximizing $I$ we force the two modules to extract as pure a version as possible of the underlying common signal.

## 2.1   The basic stereo net

We have shown how this principle can be applied to a multi-layer network that learns to extract depth from random dot stereograms (Becker and Hinton, 1992). Each network module received input from a patch of a left image and a corresponding patch of a right image, as shown in Figure 1 a). Adjacent modules received input from adjacent stereo image patches, and learned to extract depth by trying to maximize agreement between their outputs. The real-valued depth (relative to the plane of fixation) of each patch of the surface gives rise to a disparity between features in the left and right images; since that disparity is the only property that is coherent across each stereo image, the output units of modules were able to learn to accurately detect relative depth.

## 2.2   The interpolating net

The basic stereo net uses a very simple model of coherence in which an underlying parameter at one location is assumed to be approximately equal to the parameter at a neighbouring location. This model is fine for the depth of fronto-parallel surfaces but it is far from the best model of slanted or curved surfaces. Fortunately, we can use a far more general model of coherence in which the parameter at one location is assumed to be an unknown linear function of the parameters at nearby locations. The particular linear function that is appropriate can be learned by the network.

We used a network of the type shown in Figure 1 b). The depth computed locally by a module, $d_c$, was compared with the depth predicted by a linear combination $\hat{d}_c$ of the outputs of nearby modules, and the network tried to maximize the agreement between $d_c$ and $\hat{d}_c$.

The contextual prediction, $\hat{d}_c$, was produced by computing a weighted sum of the outputs of *two* adjacent modules on either side. The interpolating weights used in this sum, and all other weights in the network, were adjusted so as to maximize agreement between locally computed and contextually predicted depths. To speed the learning, we first trained the lower layers of the network as before, so that agreement was maximized between neighbouring locally computed outputs. This made it easier to learn good interpolating weights. When the network was trained on stereograms of cubic surfaces, it learned interpolating weights of $-0.147, 0.675, 0.656, -0.131$ (Becker and Hinton, 1992). Given noise free estimates of local depth, the optimal linear interpolator for a cubic surface is $-0.167, 0.667, 0.667, -0.167$.

## 3   Throwing out discontinuities

If the surface is continuous, the depth at one patch can be accurately predicted from the depths of two patches on either side. If, however, the training data contains cases in which there are depth discontinuities (see figure 2) the interpolator will also try to model these cases and this will contribute considerable noise to the interpolating weights and to the depth estimates. One way of reducing this noise is to treat the discontinuity cases as outliers and to throw them out. Rather than making a hard decision about whether a case is an outlier, we make a soft decision by using a mixture model. For each training case, the network compares the locally extracted depth, $d_c$, with the depth predicted from the nearby context, $\hat{d}_c$. It assumes that $d_c - \hat{d}_c$ is drawn from a zero-mean Gaussian if it is a continuity case and from a uniform distribution if it is a discontinuity case. It can then estimate the probability of a continuity case:

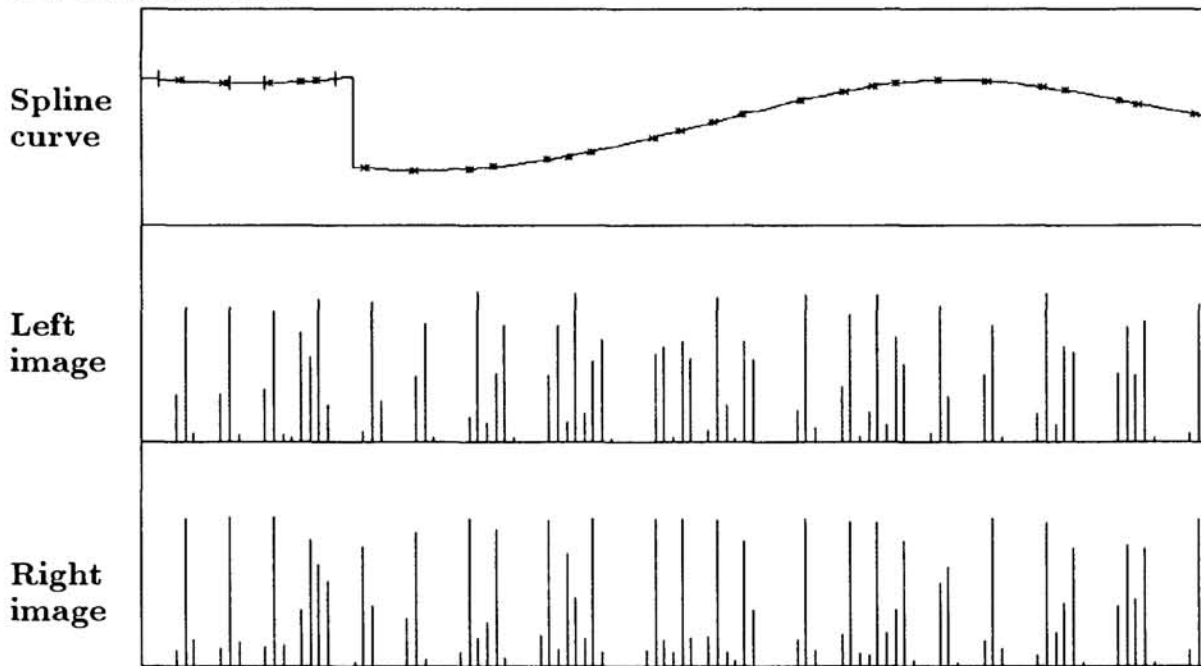

Figure 2: **Top:** *A curved surface strip with a discontinuity created by fitting 2 cubic splines through randomly chosen control points, 25 pixels apart, separated by a depth discontinuity. Feature points are randomly scattered on each spline with an average of 0.22 features per pixel.* **Bottom:** *A stereo pair of "intensity" images of the surface strip formed by taking two different projections of the feature points, filtering them through a gaussian, and sampling the filtered projections at evenly spaced sample points. The sample values in corresponding patches of the two images are used as the inputs to a module. The depth of the surface for a particular image region is directly related to the disparity between corresponding features in the left and right patch. Disparity ranges continuously from −1 to +1 image pixels. Each stereo image was 120 pixels wide and divided into 10 receptive fields 10 pixels wide and separated by 2 pixel gaps, as input for the networks shown in figure 1. The receptive field of an interpolating unit spanned 58 image pixels, and discontinuities were randomly located a minimum of 40 pixels apart, so only rarely would more than one discontinuity lie within an interpolator's receptive field.*

$$p_{cont}(d_c - \hat{d}_c) = \frac{N(d_c - \hat{d}_c, 0, \hat{V}_{cont}(d_c - \hat{d}_c))}{N(d_c - \hat{d}_c, 0, \hat{V}_{cont}(d_c - \hat{d}_c)) + k_{discont}} \tag{2}$$

where $N$ is a gaussian, and $k_{discont}$ is a constant representing a uniform density. [1]

We can now optimize the *average* information $d_c$ and $\hat{d}_c$ transmit about their common cause. We assume that no information is transmitted in discontinuity cases, so the average information depends on the probability of continuity and on the variance of $d_c + \hat{d}_c$ and $d_c - \hat{d}_c$ measured only in the continuity cases.

$$I^* = 0.5 \ P_{cont} \ \log \frac{V_{cont}(d_c + \hat{d}_c)}{V_{cont}(d_c - \hat{d}_c)} \tag{3}$$

We tried several variations of this mixture approach. The network is quite good at rejecting the discontinuity cases, but this leads to only a modest improvement in the performance of the interpolator. In cases where there is a depth discontinuity between $d_a$ and $d_b$ or between $d_d$ and $d_e$ the interpolator works moderately well because the weights on $d_a$ or $d_e$ are small. Because of the term $P_{cont}$ in equation 3 there is pressure to include these cases as continuity cases, so they probably contribute noise to the interpolating weights. In the next section we show how to avoid making a forced choice between rejecting these cases or treating them just like all the other continuity cases.

## 4   Learning a mixture of expert interpolators

The presence of a depth discontinuity somewhere within a strip of five adjacent patches does not entirely eliminate the coherence of depth across these patches. It just restricts the range over which this coherence operates. So instead of throwing out cases that contain a discontinuity, the network could try to develop a number of different, specialized interpolators each of which captures the particular type of coherence that remains in the presence of a discontinuity at a particular location. If, for example, there is a depth discontinuity between $d_c$ and $d_e$, an extrapolator with weights of $-1.0, +2.0, 0, 0$ would be an appropriate predictor of $d_c$.

Figure 3 shows the system of five expert interpolators that we used for predicting $d_c$ from the neighboring depths. To allow the system to invoke the appropriate interpolator, each expert has its own "controller" which must learn to detect the presence of a discontinuity at a particular location (or the absence of a discontinuity in the case of the interpolator for pure continuity cases). The outputs of the controllers are normalized, as shown in figure 3, so that they form a probability distribution. We can think of these normalized outputs as the probability with which the system selects a particular expert. The controllers get to see all five local depth estimates and most of them learn to detect particular depth discontinuities by using large weights of opposite sign on the local depth estimates of neighboring patches.

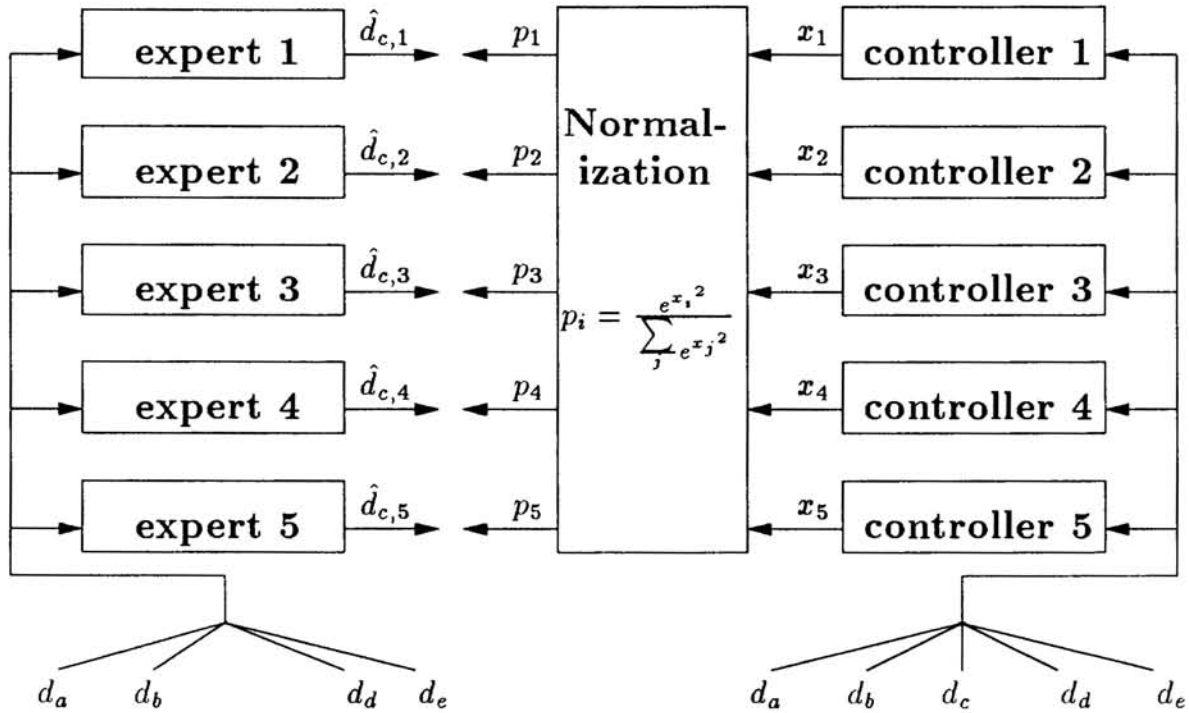

Figure 3: *The architecture of the mixture of interpolators and discontinuity detectors. We actually used a larger modular network and equality constraints between modules, as described in figure 1 b), with 6 copies of the architecture shown here. Each copy received input from different but overlapping parts of the input.*

Figure 4 shows the weights learned by the experts and by their controllers. As expected, there is one interpolator (the top one) that is appropriate for continuity cases and four other interpolators that are appropriate for the four different locations of a discontinuity. In interpreting the weights of the controllers it is important to remember that a controller which produces a small $x$ value for a particular case may nevertheless assign high probability to its expert if all the other controllers produce even smaller $x$ values.

## 4.1   The learning procedure

In the example presented here, we first trained the network shown in figure 1b) on images with discontinuities. We then used the outputs of the depth extracting layer, $d_a, \ldots, d_e$ as the inputs to the expert interpolators and their controllers. The system learned a set of expert interpolators without backpropagating derivatives all the way down to the weights of the local depth extracting modules. So the local depth estimates $d_a, \ldots, d_e$ did not change as the interpolators were learned.

To train the system we used an unsupervised version of the competing experts algorithm described by Jacobs, Jordan, Nowlan and Hinton (1991). The output of the $i^{th}$ expert, $\hat{d}_{c,i}$, is treated as the mean of a Gaussian distribution with variance $\sigma^2$ and the normalized output of each controller, $p_i$, is treated as the mixing proportion of that Gaussian. So, for each training case, the outputs of the experts and their controllers define a probability distribution that is a mixture of Gaussians. The aim

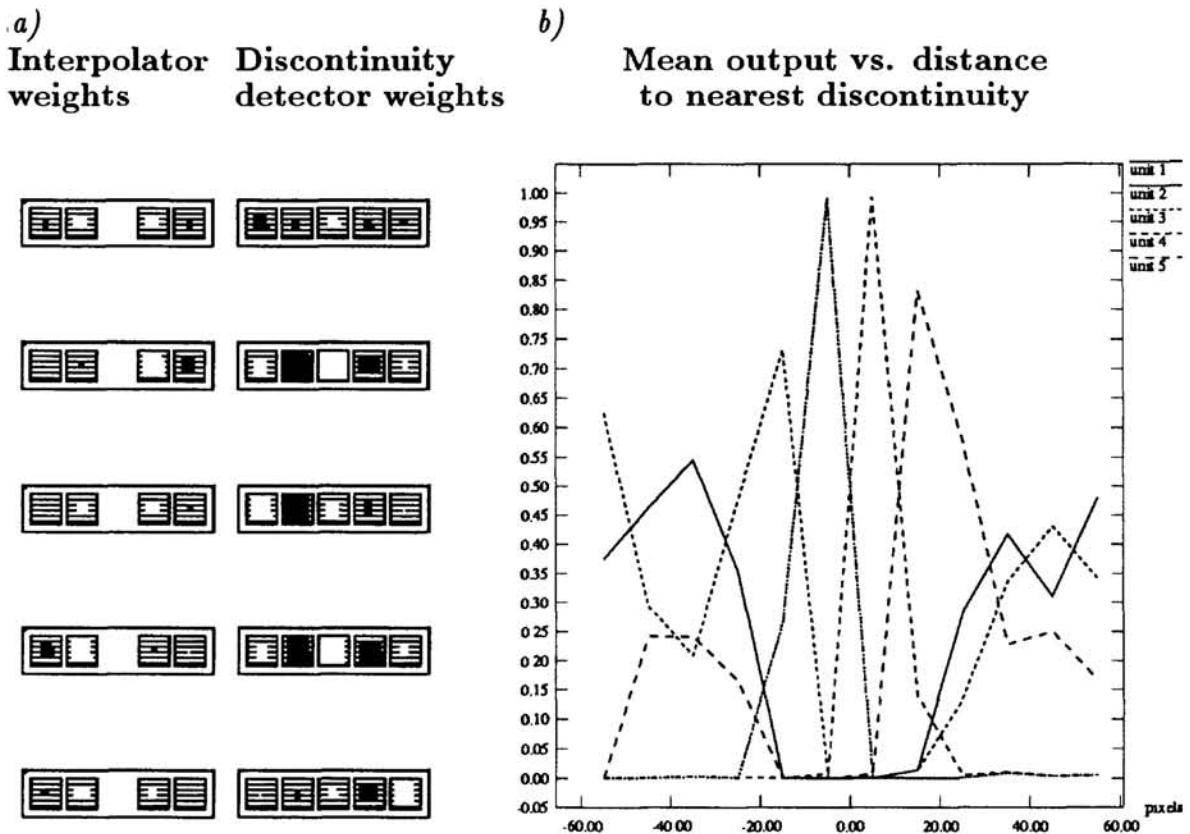

*a)*
**Interpolator    Discontinuity**
**weights          detector weights**

*b)*
**Mean output vs. distance**
**to nearest discontinuity**

Figure 4: *a) Typical weights learned by the five competing interpolators and corresponding five discontinuity detectors. Positive weights are shown in white, and negative weights in black. b)The mean probabilities computed by each discontinuity detector are plotted against the the distance from the center of the units' receptive field to the nearest discontinuity. The probabilistic outputs are averaged over an ensemble of 1000 test cases. If the nearest discontinuity is beyond ± thirty pixels, it is outside the units' receptive field and the case is therefore a continuity example.*

of the learning is to maximize the log probability density of the desired output, $d_c$, under this mixture of Gaussians distribution. For a particular training case this log probability is given by:

$$\log P(d_c) = \log \sum_i p_i \frac{1}{\sqrt{2\pi}\sigma} \exp\left(-\frac{(d_c - \hat{d}_{c,i})^2}{2\sigma^2}\right) \tag{4}$$

By taking derivatives of this objective function we can simultaneously learn the weights in the experts and in the controllers. For the results shown here, the nework was trained for 30 conjugate gradient iterations on a set of 1000 random dot stereograms with discontinuities.

The rationale for the use of a variance ratio in equation 1 is to prevent the variances of $d_a$ and $d_b$ collapsing to zero. Because the local estimates $d_1, \ldots, d_5$ did not change as the system learned the expert interpolators, it was possible to use $(d_c - \hat{d}_{c,i})^2$ in the objective function without worrying about the possibility that the variance of $d_c$ across cases would collapse to zero during the learning. Ideally we would like to

refine the weights of the local depth estimators to maximize their agreement with the contextually predicted depths produced by the mixture of expert interpolators. One way to do this would be to generalize equation 3 to handle a mixture of expert interpolators:

$$I^* = 0.5 \sum_i P_i \, \log \frac{V_i(d_c + \hat{d}_{c,i})}{V_i(d_c - \hat{d}_{c,i})} \tag{5}$$

Alternatively we could modify equation 4 by normalizing the difference $(d_c - \hat{d}_{c,i})^2$ by the actual variance of $d_c$, though this makes the derivatives considerably more complicated.

## 5  Discussion

The competing controllers in figure 3 explicitly represent which regularity applies in a particular region. The outputs of the controllers for nearby regions may themselves exhibit coherence at a larger spatial scale, so the same learning technique could be applied recursively. In 2-D images this should allow the continuity of depth edges to be discovered.

The approach presented here should be applicable to other domains which contain a mixture of alternative local regularities across space or time. For example, a rigid shape causes a linear constraint between the locations of its parts in an image, so if there are many possible shapes, there are many alternative local regularities (Zemel and Hinton, 1991).

Our learning procedure differs from methods that try to capture as much information as possible about the input (Linsker, 1988; Atick and Redlich, 1990) because we ignore information in the input that is not coherent across space.

### Acknowledgements

This research was funded by grants from NSERC and the Ontario Information Technology Research Centre. Hinton is Noranda fellow of the Canadian Institute for Advanced Research. Thanks to John Bridle and Steve Nowlan for helpful discussions.

## Footnotes

[1]We empirically select a good (fixed) value of $k_{discont}$, and we choose a starting value of $\hat{V}_{cont}(d_c - \hat{d}_c)$ (some proportion of the initial variance of $d_c - \hat{d}_c$), and gradually shrink it during learning.

### References

Atick, J. J. and Redlich, A. N. (1990). Towards a theory of early visual processing. Technical Report IASSNS-HEP-90/10, Institute for Advanced Study, Princeton.

Becker, S. and Hinton, G. E. (1992). A self-organizing neural network that discovers surfaces in random-dot stereograms. January 1992 *Nature*.

Jacobs, R. A., Jordan, M. I., Nowlan, S. J., and Hinton, G. E. (1991). Adaptive mixtures of local experts. *Neural Computation*, 3(1).

Linsker, R. (1988). Self-organization in a perceptual network. *IEEE Computer*, March, 21:105–117.

Zemel, R. S. and Hinton, G. E. (1991). Discovering viewpoint-invariant relationships that characterize objects. In *Advances In Neural Information Processing Systems 3*, pages 299–305. Morgan Kaufmann Publishers.